# Learning transport operators for image manifolds

**Benjamin J. Culpepper**
Department of EECS
Computer Science Division
University of California, Berkeley
Berkeley, CA 94720
bjc@cs.berkeley.edu

**Bruno A. Olshausen**
Helen Wills Neuroscience Institute
& School of Optometry
University of California, Berkeley
Berkeley, CA 94720
baolshausen@berkeley.edu

## Abstract

We describe an unsupervised manifold learning algorithm that represents a surface through a compact description of operators that traverse it. The operators are based on matrix exponentials, which are the solution to a system of first-order linear differential equations. The matrix exponents are represented by a basis that is adapted to the statistics of the data so that the infinitesimal generator for a trajectory along the underlying manifold can be produced by linearly composing a few elements. The method is applied to recover topological structure from low dimensional synthetic data, and to model local structure in how natural images change over time and scale.

## 1   Introduction

It is well known that natural images occupy a small fraction of the space of all possible images. Moreover, as images change over time in response to observer motion or changes in the environment they trace out particular trajectories along manifolds in this space. It is reasonable to expect that perceptual systems have evolved ways to efficiently model these manifolds, and thus mathematical models that capture their structure in operators that transport along them may be of use for understanding perceptual systems, as well as for engineering artificial vision systems. In this paper, we derive methods for learning these transport operators from data.

Rather than simply learning a mapping of individual data points to a low-dimensional space, we seek a compact representation of the entire manifold via the operators that traverse it. We investigate a direct application of the Lie approach to invariance [1] utilizing a matrix exponential generative model for transforming images. This is in contrast to previous methods that rely mainly upon a first-order Taylor series approximation of the matrix exponential [2,3], and bilinear models, in which the transformation variables interact multiplicatively with the input [4,5,6]. It is also distinct from the class of methods that learn embeddings of manifolds from point cloud data [7,8,9,10]. The spirit of this work is similar to [11], which also uses a spectral decomposition to make learning tractable in extremely high dimensional Lie groups, such as those over images. We share the goal of [12] of learning a model of the manifold which can then be generalized to new data.

Here we show how a particular class of transport operators for moving along manifolds may be learned from data. The model is first applied to synthetic datasets to demonstrate interesting cases where it can recover topology, and that for more difficult cases it neatly approximates the local structure. Subsequently, we apply it to time-varying natural images and extrapolate along inferred trajectories to demonstrate super-resolution and temporal filling-in of missing video frames.

## 2 Problem formulation

Let us consider an image of the visual world at time $t$ as a point $\mathbf{x} \in \mathcal{R}^N$, where the elements of $\mathbf{x}$ correspond to image pixels. We describe the evolution of $\mathbf{x}$ as

$$\dot{\mathbf{x}} = \mathbf{A}\,\mathbf{x}\,, \tag{1}$$

where the matrix $\mathbf{A}$ is a linear operator capturing some action in the environment that transforms the image. Such an action belongs to a family that occupies a subspace of $\mathcal{R}^{N \times N}$ given by

$$\mathbf{A} = \sum_{m=1}^{M} \mathbf{\Psi}_m\, c_m \tag{2}$$

for some $M \leq N^2$ (usually $M << N^2$), with $\mathbf{\Psi}_m \in \mathcal{R}^{N \times N}$. The amount of a particular action from the dictionary $\mathbf{\Psi}_m$ that occurs is controlled by the corresponding $c_m$. At $t = 0$, a vision system takes an image $\mathbf{x}_0$, and then makes repeated observations at intervals $\Delta t$. Given $\mathbf{x}_0$, the solution to (1) traces out a continuously differentiable manifold of images given by $\mathbf{x}_t = \exp(\mathbf{A}t)\,\mathbf{x}_0$, which we observe periodically. Our goal is to learn an appropriate set of bases, $\mathbf{\Psi}$, that allow for a compact description of this set of transformations by training on many pairs of related observations.

This generative model for transformed images has a number of attractive properties. First, it factors apart the time-varying image into an invariant part (the initial image, $\mathbf{x}_0$) and variant part (the transformation, parameterized by the coefficient vector $\mathbf{c}$), thus making explicit the underlying causes. Second, the learned exponential operators are quite powerful in terms of modeling capacity, compared to their linear counterparts. Lastly, the partial derivatives of the objective function have a simple form that may be computed efficiently.

## 3 Algorithm

The model parameters are learned by maximizing the log-likelihood of the model. Consider two 'close' states of the system in isolation. Let $\mathbf{x}_0$ be our initial condition, and $\mathbf{x}_1$ be a second observation. These points are related through an exponentiated matrix that itself is composed of a few basis elements, plus zero-mean white i.i.d. Gaussian noise, $\mathbf{n}$:

$$\mathbf{x}_1 = \mathbf{T}(\mathbf{c})\,\mathbf{x}_0 + \mathbf{n} \tag{3}$$

$$\mathbf{T}(\mathbf{c}) = \exp(\sum_m \mathbf{\Psi}_m\, c_m)\,. \tag{4}$$

We assume a factorial sparse prior over the transform variables $\mathbf{c}$ of the form $P(c_m) \propto \exp(-\zeta\,|c_m|)$. The negative log of the posterior probability of the data under the model is given by

$$E = \frac{1}{2}||\mathbf{x}_1 - \mathbf{T}(\mathbf{c})\,\mathbf{x}_0||_2^2 + \frac{\gamma}{2}\sum_m ||\mathbf{\Psi}_m||_{\mathrm{F}}^2 + \zeta||\mathbf{c}||_1\,, \tag{5}$$

where $||\cdot||_{\mathrm{F}}$ is the Frobenius norm, which acts to regularize the dictionary element lengths. The 1-norm encourages sparsity. Given two data points, the solution of the $\mathbf{c}$ variables which relate them through $\mathbf{\Psi}$ is found by a fast minimization of $E$ with respect to $\mathbf{c}$.

Learning of the basis $\mathbf{\Psi}$ proceeds by gradient descent with respect to $E$. (Note that this constitutes a variational approximation to the log-likelihood, similar to [13].) The $\mathbf{\Psi}$ variables are initialized randomly, and adjusted according to $\Delta\mathbf{\Psi} = -\eta\frac{\partial E}{\partial \mathbf{\Psi}}$, using the solution, $\mathbf{c}$, for a pair of observations $\mathbf{x}_0, \mathbf{x}_1$. Figure 1 outlines the steps of the algorithm.

The partial derivatives of $E$ w.r.t. $\mathbf{c}$ and $\mathbf{\Psi}$ can be cast in a simple form using the spectral decomposition of $\mathbf{A}$, given by $\sum_\alpha \lambda_\alpha \mathbf{u}_\alpha \mathbf{v}_\alpha^{\mathrm{T}}$, with right eigenvectors $\mathbf{u}_\alpha$, left eigenvectors $\mathbf{v}_\alpha$, and eigenvalues $\lambda_\alpha$ [14]. Let $\mathbf{U} = [\mathbf{u}_1\mathbf{u}_2...\mathbf{u}_N]$, $\mathbf{V} = [\mathbf{v}_1\mathbf{v}_2...\mathbf{v}_N]$ and $\mathbf{D}$ be a diagonal matrix of the eigenvalues $\lambda_\alpha$. Then

$$\frac{\partial \exp(A)_{ij}}{\partial A_{kl}} = \sum_{\alpha\beta} F_{\alpha\beta} U_{i\alpha} V_{k\alpha} U_{l\beta} V_{j\beta}\,, \tag{6}$$

```
1   choose $M \leq N^2$
2   initialize $\mathbf{\Psi}$
3   while stopping criteria is not met,
4       pick $\mathbf{x}_0, \mathbf{x}_1$
5       initialize $\mathbf{c}$ to zeros
6       $\mathbf{c} \leftarrow \arg\min_{\mathbf{c}} E$
7       $\Delta\mathbf{\Psi} = -\eta \frac{\partial E}{\partial \mathbf{\Psi}}$
8       sort $\mathbf{\Psi}_m$ by $||\mathbf{\Psi}_m||_{\mathrm{F}}$
9       $M \leftarrow \max m$ s.t. $||\mathbf{\Psi}_m||_{\mathrm{F}} > \epsilon$
```

Figure 1: **Pseudo-code for the learning algorithm.** Steps 1-2 initialize. A typical stopping criteria in step 3 is that the reconstruction error or sparsity on some held-out data falls below a threshold. Steps 4-6 compute an E-step on some pair of data points. Step 7 computes a 'partial' M-step. Steps 8-9 shrink the subspace spanned by the dictionary if one or more of the elements have shrunk sufficiently in norm.

where the matrix $\mathbf{F}$ is given by:

$$
F_{\alpha\beta} = \begin{cases} \frac{\exp(\lambda_\beta) - \exp(\lambda_\alpha)}{\lambda_\beta - \lambda_\alpha} & \text{if} \quad \lambda_\beta \neq \lambda_\alpha \\ \exp(\lambda_\alpha) & \text{otherwise} \end{cases} \tag{7}
$$

Application of the chain rule and a re-arrangement of terms yields simplified forms for the partials of $E$ w.r.t. $\mathbf{c}$ and $\mathbf{\Psi}$. After computing two intermediate terms $\mathbf{P}$ and $\mathbf{Q}$,

$$
\mathbf{P} = \mathbf{U}^{\mathrm{T}}(\mathbf{x_1}\mathbf{x_0}^{\mathrm{T}} + \mathbf{x_0}\mathbf{x_0}^{\mathrm{T}}\mathbf{T}^{\mathrm{T}})\mathbf{V} \tag{8}
$$

$$
Q_{kl} = \sum_{\alpha\beta} V_{k\alpha} U_{l\beta} F_{\alpha\beta} P_{\alpha\beta}, \tag{9}
$$

the two partial derivatives for inference and learning are:

$$
\frac{\partial E}{\partial c_m} = \sum_{kl} Q_{kl}\,\Psi_{klm} + \zeta\,\mathrm{sgn}(c_m) \tag{10}
$$

$$
\frac{\partial E}{\partial \Psi_{klm}} = Q_{kl}\,c_m + \gamma\,\Psi_{klm}. \tag{11}
$$

The order of complexity for both derivatives is determined by the computation of $\mathbf{Q}$, which requires an eigen-decomposition and a few matrix multiplications, giving $O(N^p)$ with $2 < p < 3$.

## 4   Experiments on point sets

We first test the model by applying it to simple datasets where the solutions are known: learning the topology of a sphere and a torus. Second, we apply the model to learn the manifold of time-varying responses to a natural movie from complex oriented filters. These demonstrations illustrate the algorithm's capability for learning significant non-linear structure.

We have also applied the model to the Klein bottle. Though closely related to the torus, it is an example of a low-dimensional surface whose topology can not be captured by a first-order Lie operator, though our model is able to interpolate between points on the surface using a piecewise approximation (see the supplementary material accompanying this paper for further discussion of this point).

Related pairs of points on a torus are generated by choosing two angles $\theta_0, \phi_0$ uniformly at random from $[0, 2\pi]$; two related angles $\theta_1, \phi_1$ are produced by sampling from two von Mises distributions with means $\theta_0$ and $\phi_0$, and concentration $\kappa = 5$ using the circular statistics toolbox of [15]. For the sphere, we generate the first pair of angles using the normal-deviate method, to avoid concentration of samples near the poles. Though parameterized by two angles, the coordinates of points on these surfaces are 3- and 4-dimensional; pairs of points $\mathbf{x}_t$ for $t = 0, 1$ on the unit sphere are given by $\mathbf{x}_t = (\sin\theta_t \cos\phi_t, \sin\theta_t \sin\phi_t, \cos\theta_t)$, and points on a torus by $\mathbf{x}_t = (\cos\theta_t, \sin\theta_t, \cos\phi_t, \sin\phi_t)$.

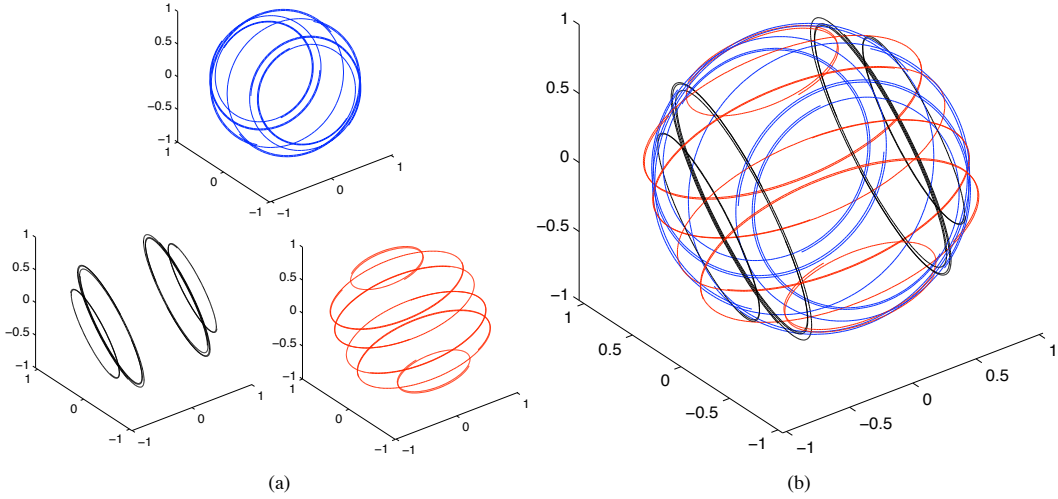

<div align="center">(a)                                   (b)</div>

Figure 2: **Orbits of learned sphere operators.** (a) Three $\boldsymbol{\Psi}_m$ basis elements applied to points at the six poles of the sphere, $(1,0,0), (0,1,0), (0,0,1), (-1,0,0), (0,-1,0),$ and $(0,0,-1)$. The orbits are generated by setting $\mathbf{x}_0$ to a pole, then plotting $\mathbf{x}_t = \exp(\boldsymbol{\Psi}_m\, t)\, \mathbf{x}_0$ for $t = [-100, 100]$. (b) When superimposed on top of each other, the three sets of orbits clearly define the surface of a sphere.

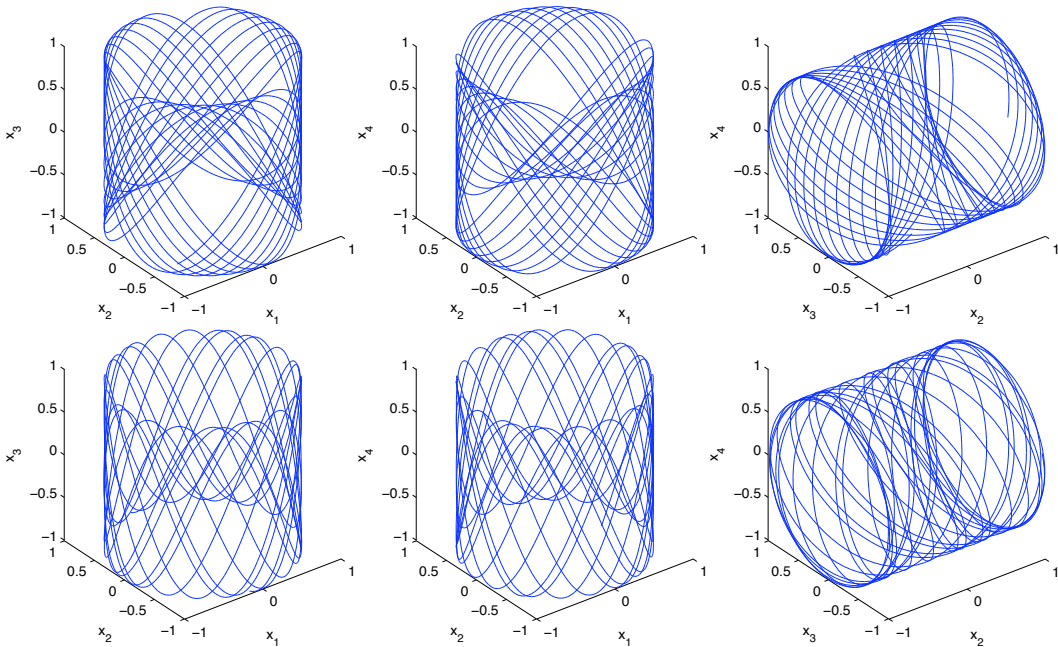

Figure 3: **Orbits of learned torus operators.** Each row shows three projections of a $\boldsymbol{\Psi}_m$ basis element applied to a point on the surface of the torus. The orbits shown are generated by setting $\mathbf{x}_0 = (0,1,0,1)$ then plotting $\mathbf{x}_t = \exp(\boldsymbol{\Psi}_m\, t)\, \mathbf{x}_0$ for $t = [-1000, 1000]$ in projections constructed from each triplet of the four coordinates. In each plot, two coordinates always obey a circular relationship, while the third varies more freely.

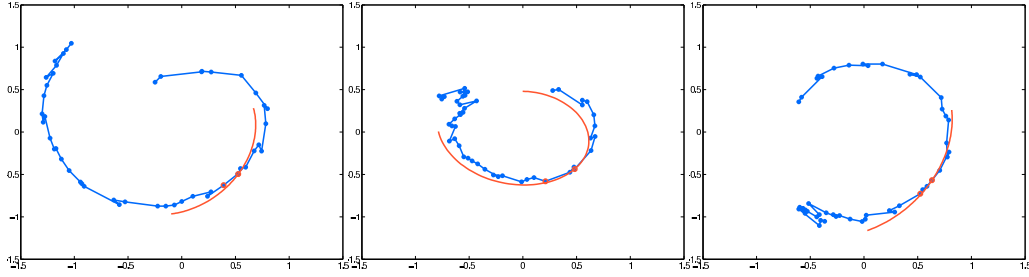

Figure 4: **Learning transformations of oriented filter pairs across time.** The orbits of three three complex filter outputs in response to a natural movie. The blue points denote the complex output for each frame in the movie sequence and are linked to their neighbors via the blue line. The points circled in red were observed by the model, and the red curve shows an extrapolation along the estimated trajectory.

For the sphere, $N = 3$, thus setting $M = 9$ gives the model the freedom to generate the full space of $\mathbf{A}$ operators. The $\mathbf{\Psi}$ are initialized to mean-zero white Gaussian noise with variance 0.01, and 10, 000 learning updates are computed by generating a pair of related points, minimizing $E$ w.r.t. $\mathbf{c}$, then updating $\mathbf{\Psi}$ according to $\Delta\mathbf{\Psi} = -\eta \frac{\partial E}{\partial \mathbf{\Psi}}$. In all of the point set experiments, $\gamma = 0.0001$ and $\zeta = 0.01$. For cases where topology can be recovered, the solution is robust to the settings of $\gamma$ and $\zeta$ – changing either variable by an order of magnitude does not change the solution, though it may increase the number of learning steps required to get to it. In cases where the topology can not be recovered, the influence on the solution of the settings of $\gamma$ and $\zeta$ is more subtle, as their relative values effectively trade-off the importance of data reconstruction and the sparsity of the vector $\mathbf{c}$. We adjust $\eta$ during learning as follows: when $\Delta\mathbf{\Psi}$ causes $E$ to decrease, we multiply $\eta$ by 1.01; otherwise, we multiply $\eta$ by 0.99. When the model has more parameters than it needs to fully capture the topology of the sphere this fact is evident from the solution it learns: six of the dictionary elements $\mathbf{\Psi}_m$ drop out (they have norm less than $10^{-6}$), since the F-norm 'weight decay' term kills off dictionary elements that are used rarely. Figure 2 shows orbits produced by applying each of the remaining $\mathbf{\Psi}_m$ operators to points on the sphere. Similar experiments are successful for the torus; Figure 3 shows trajectories of the operators learned for the torus.

As an intermediate step towards modeling time varying natural images, we investigate the model's ability to learn the response surface for a single complex oriented filter to a moving image. A complex pyramid is built from each frame in a movie, and pairs of filter responses 1 to 4 frames apart are observed by the model. Four 2x2 basis functions are learned in the manner described above. Figure 4 shows three representative examples that illustrate how well the model is able to extrapolate from the solution estimated using the learned basis $\mathbf{\Psi}$, and complex responses from the same filter within a 4 frame time interval. In most cases, this trajectory follows the data closely for several frames.

## 5   Experiments on movies

In the image domain, our model has potential applications in temporal interpolation/filling in of video, super-resolution, compression, and geodesic distance estimation. We apply the model to moving natural images and investigate the first three applications; the third will be the subject of future work. Here we report on the ability of the model to learn transformations across time, as well as across scales of the Laplacian pyramid. Our data is many short grayscale video sequences of Africa from the BBC.

### 5.1   Time

We apply the model to natural movies by presenting it with patches of adjacent frame pairs. Using an analytically generated infinitesimal shift operator, we first run a series of experiments to determine the effect of local minima on the recovery of a known displacement through the minimization of $E$

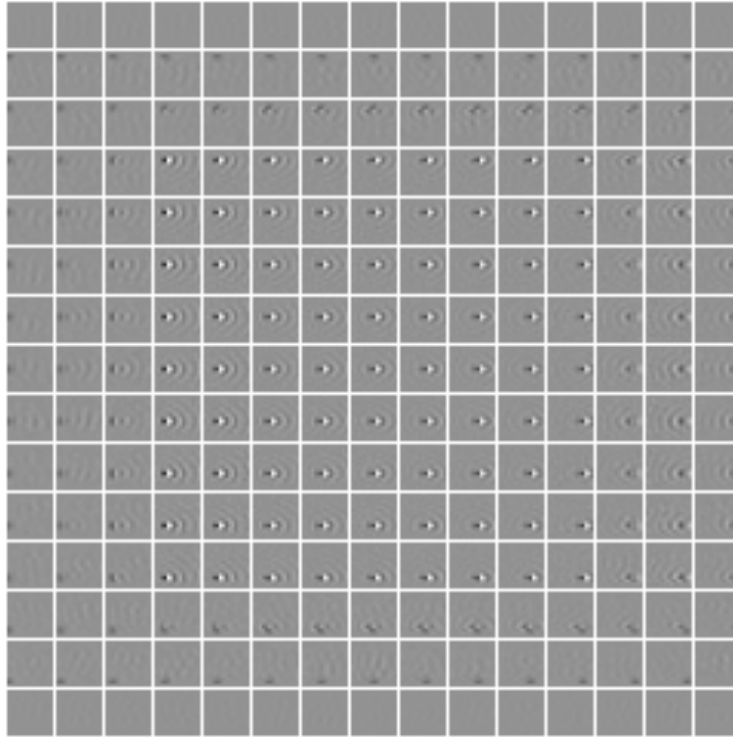

Figure 5: **Shift operator learned from synthetically transformed natural images.** The operator $\mathbf{\Psi}_1$, displayed as an array of weights that, for each output pixel, shows the strength of its connection to each input pixel. Each of the 15x15 arrays represents one output pixel's connections. Because of the $1/f^2$ falloff in the power spectrum of natural images, synthetic images with a wildly different distribution of spatial frequency content, such as uncorrelated noise, will not be properly shifted by this operator.

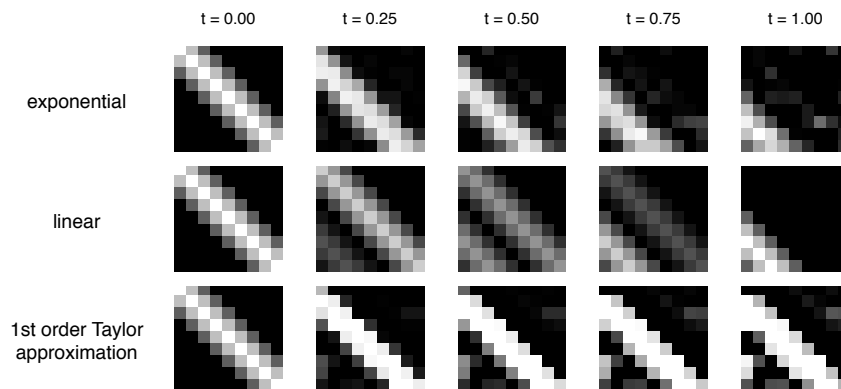

Figure 6: **Interpolating between shifted images to temporally fill in missing video frames.** Two images $\mathbf{x}_0$ and $\mathbf{x}_1$ are generated by convolving an image of a diagonal line and a shifted diagonal line by a 3x3 Gaussian kernel with $\sigma = 0.8$, and the operator $\mathbf{A}$ is inferred. The top row shows the sequence of images $\mathbf{x}_t = \exp(\mathbf{A}t)\mathbf{x}_0$ for $t = 0.25, 0.50, 0.75, 1.00$. The middle row shows linear interpolation between $\mathbf{x}_0$ and $\mathbf{x}_1$. The bottom row shows the sequence of images $\mathbf{x}_t = (\mathbf{I} + \mathbf{A}t)\mathbf{x}_0$, that is, the first-order Taylor expansion of the matrix exponential, which performs poorly for shifts greater than one pixel.

w.r.t. $\mathbf{c}$. When initialized to zero, the $\mathbf{c}$ vector often converges to the wrong displacement, but this problem can be avoided with high probability using a coarse-to-fine technique [16,17]. Doing so requires a slight alteration to our inference algorithm: now we must solve a sequence of optimization problems on frame pairs convolved with a Gaussian kernel whose variance is progressively decreased. At each step in the sequence, both frames are convolved by the kernel before a patch is selected. For the first step, the $\mathbf{c}$ variables are initialized to zero; for subsequent steps they are initialized to the solution of the previous step. For our analytical shifting operator, two blurring filters – first a 5x5 kernel with variance 10, then a 3x3 kernel with variance 5 – reliably gives a proper initialization for the final minimization that runs on the unaltered data.

For control purposes, the video for this experiment comes from a camera fly over; thus, most of the motion in the scene is due to camera motion. We apply the model pairs of 11x11 patches, selected from random locations in the video, but discarding patches near the horizon where there is little or no motion. We initialize $M = 16$; after learning, the basis function with the longest norm has the structure of a shift operator in the primary direction of motion taking place in the video sequence. Using these 16 operators, we run inference on 1,000 randomly selected pairs of patches from a second video, not used during learning, and measure the quality of the reconstruction as the trajectory is used to predict into the future. At 5 frames into the future, our model is able to maintain an average SNR of 7, compared to SNR 5 when a first-order Taylor approximation is used in place of the matrix exponential; for comparison, the average SNR for the identity transformation model on this data is 1.

Since the primary form of motion going on in these small patches is translation, we also train a single operator using artificially translated natural data to make clear that the model can learn this case completely. For this last experiment we take a 360x360 pixel frame of our natural movie, and continuously translate the entire frame in the Fourier domain by a displacement chosen uniformly at random from $[0, 3]$ pixels. We then randomly select a 15x15 region on the interior of the pair of frames and use the two 225 pixel vectors as our $\mathbf{x}_0$ and $\mathbf{x}_1$. We modify the objective function to be

$$E = \frac{1}{2}||\mathbf{W}\left[\mathbf{x}_1 - \mathbf{T}(\mathbf{c})\,\mathbf{x}_0\right]||_2^2 + \frac{\gamma}{2}\sum_m ||\mathbf{\Psi}_m||_{\mathrm{F}}^2 + \zeta||\mathbf{c}||_1\,, \tag{12}$$

where $\mathbf{W}$ is a binary windowing function that selects the central 9x9 region from a 15x15 patch; thus, the residual errors that come from new content translating into the patch are ignored. After learning, the basis function $\mathbf{\Psi}_1$ (shown in figure 5) is capable of translating natural images up to 3 pixels while maintaining an average SNR of 16 in the 9x9 center region. Figure 6 shows how this operator is able to correctly interpolate between two measurements of a shifted image in order to temporally up-sample a movie.

## 5.2 Scale

The model can also learn to transform between successive scales in the Laplacian pyramid built from a single frame of a video sequence. Figure 7 depicts the system transforming an image patch from scale 2 to 3 of a 256x256 pixel image. We initialize $M = 100$, but many basis elements shrink during learning; we use only the 16 $\mathbf{\Psi}_m$ with non-negligible norm to encode a scale change. The basis $\mathbf{\Psi}$ is initialized to mean-zero white Gaussian noise with variance 0.01; the same inference and learning procedures as described for the point sets are then run on pairs $\mathbf{x}_0, \mathbf{x}_1$ selected at random from the corpus of image sequences in the following way. First, we choose a random frame from a random sequence, then up-sample and blur scale 3 of its Laplacian pyramid. Second, we select an 8x8 patch from scale 2 ($\mathbf{x}_0$) of the corresponding up-blurred image patch ($\mathbf{x}_1$). Were it not for the highly structured manifold on which natural images live, the proposition of finding an operator that maps a blurred, subsampled image to its high-resolution original state would seem untenable. However, our results show that in many cases, a reduced representation of such two-way mappings can be found, even for small patches.

## 6   Discussion and conclusions

We have shown that it is possible to learn low-dimensional parameterizations of operators that transport along non-linear manifolds formed by natural images, both across time and scale. Our focus thus far has been primarily on understanding the model and how to properly optimize its parameters,

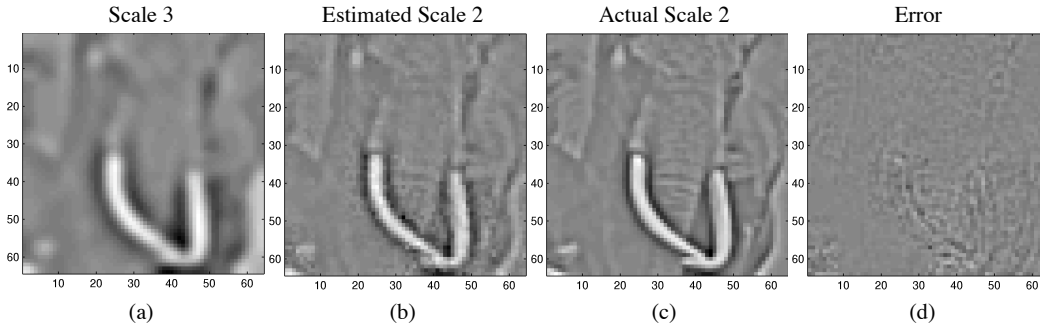

| Scale 3 | Estimated Scale 2 | Actual Scale 2 | Error |
|:---:|:---:|:---:|:---:|
| (a) | (b) | (c) | (d) |

Figure 7: **Learning transformations across scale.** (a) Scale 3 of the Laplacian pyramid for a natural scene we wish to code, by describing how it transforms across scale, in terms of our learned dictionary. (b) The estimated scale 2, computed by transforming 8x8 regions of the up-sampled and blurred scale 3. The estimated scale 2 has SNR 9.60; (c) shows the actual scale 2 and (d) shows the errors made by our estimation. For reconstruction we use only 16 dictionary elements.

as little work has previously been done on learning such high dimensional Lie groups. A promising direction for future work is to explore higher-order models capable of capturing non-commutative operators, such as

$$\mathbf{x}_t = \exp(\mathbf{\Psi}_1 c_1) \exp(\mathbf{\Psi}_2 c_2) \cdots \exp(\mathbf{\Psi}_K c_K) \mathbf{x}_0 , \qquad (13)$$

as this formulation may be more parsimonious for factoring apart transformations which are prevalent in natural movies, such as combinations of translation and rotation.

Early attempts to model the manifold structure of images train on densely sampled point clouds and find an embedding into a small number of coordinates along the manifold. However such an approach does not actually constitute a model, since there is no function for mapping arbitrary points, or moving along the manifold. One must always refer back to original data points on which the model was trained – i.e., it works as a lookup table rather than being an abstraction of the data. Here, by learning operators that transport along the manifold we have been able to learn a compact description of its structure.

This model-based representation can be leveraged to compute geodesics using a numerical approximation to the arc length integral:

$$S = \int_0^1 ||\mathbf{A} \exp(\mathbf{A}\,t)||_2^2 \, \mathrm{d}t = \lim_{T \to \infty} \sum_{t=1}^{T} || \exp(\mathbf{A}\,\frac{t}{T})\,\mathbf{x}_0 - \exp(\mathbf{A}\,\frac{t-1}{T})\,\mathbf{x}_0||_2^2 , \qquad (14)$$

where $T$ is the number of segments chosen to use in the piecewise linear approximation of the curve, and each term in the summation gives the length of a segment. We believe that this aspect of our model will be of use in difficult classification problems, such as face identification, where Euclidean distances measured in pixel-space give poor results.

Previous attempts to learn Lie group operators have focused on linear approximations. Here we show that utilizing the full Lie operator/matrix exponential in learning, while computationally intensive, is tractable, even in the extremely high dimensional cases required by models of natural movies. Our spectral decomposition is the key component that enables this, and, in combination with careful mitigation of local minima in the objective function using a coarse-to-fine technique, gives us the power to factor out large transformations from data.

One shortcoming of the approach described here is that transformations are modeled in the original pixel domain. Potentially these transformations may be described more economically by working in a feature space, such as a sparse decomposition of the image. This is a direction of ongoing work.

## Acknowledgments

The authors gratefully acknowledge many useful discussions with Jascha Sohl-Dickstein, Jimmy Wang, Kilian Koepsell, Charles Cadieu, and Amir Khosrowshahi, and the insightful comments from our anonymous reviewers.

# References

[1] VanGool, L., Moons, T., Pauwels, E. & Oosterlinck, A. (1995) Vision and Lie's approach to invariance. *Image and Vision Computing*, 13(4): 259-277.

[2] Miao, X. & Rao, R.P.N. (2007) Learning the Lie groups of visual invariance. *Neural Computation*, 19(10): 2665-2693.

[3] Rao, R.P.N & Ruderman D.L. (1999) Learning Lie Groups for Invariant Visual Perception. *Advances in Neural Information Processing Systems*, 11:810-816. Cambridge, MA: MIT Press.

[4] Grimes, D.B., & Rao, R.P.N. (2002). A Bilinear Model for Sparse Coding. *Advances in Neural Information Processing Systems*, 15. Cambridge, MA: MIT Press.

[5] Olshausen, B.A., Cadieu, C., Culpepper, B.J. & Warland, D. (2007) Bilinear Models of Natural Images. *SPIE Proceedings vol. 6492: Human Vision Electronic Imaging XII (B.E. Rogowitz, T.N. Pappas, S.J. Daly, Eds.)*, Jan 28 - Feb 1, 2007, San Jose, California.

[6] Tenenbaum, J. B. & Freeman, W. T. (2000) Separating style and content with bilinear models. *Neural Computation*, 12(6):1247-1283.

[7] Roweis, S. & Saul, L. (2000) Nonlinear dimensionality reduction by locally linear embedding. *Science*, 290(5500): 2323-2326.

[8] Weinberger, K. Q. & Saul, L. K. (2004) Unsupervised learning of image manifolds by semidefinite programming. *Computer Vision and Pattern Recognition*.

[9] Tenenbaum, J. B., de Silva, V. & Langford, J. C. (2000) A Global Geometric Framework for Nonlinear Dimensionality Reduction. *Science*, 22 December 2000: 2319-2323.

[10] Belkin, M., & Niyogi, P. (2002). Laplacian eigenmaps and spectral techniques for embedding and clustering. *Advances in Neural Information Processing Systems*, 14. Cambridge, MA: MIT Press.

[11] Wang, C. M., Sohl-Dickstein, J., & Olshausen, B. A. (2009) Unsupervised Learning of Lie Group Operators from Natural Movies. *Redwood Center for Theoretical Neuroscience, Technical Report*; RCTR 01-09.

[12] Dollar, P., Rabaud, V., & Belongie, S (2007) Non-isometric Manifold Learning: Analysis and an Algorithm. *Int. Conf. on Machine Learning* , 241-248.

[13] Olshausen, B.A. & Field, D.J. (1997) Sparse Coding with an Overcomplete Basis Set: A Strategy Employed by V1? *Vision Research*, 37: 3311-3325.

[14] Ortiz, M., Radovitzky, R.A. & Repetto, E.A (2001) The computation of the exponential and logarithmic mappings and their first and second linearizations. *International Journal For Numerical Methods In Engineering.* 52: 1431-1441.

[15] Berens, P. & Velasco, M. J. (2009) The circular statistics toolbox for Matlab. *MPI Technical Report*, 184.

[16] Anandan, P. (1989) A computational framework and an algorithm for the measurement of visual motion. *Int. J. Comput. Vision*, 2(3): 283-310.

[17] Glazer, F. (1987) Hierarchical Motion Detection. *Ph.D. thesis, Univ. of Massachusetts, Amherst, MA*; COINS TR 87-02.

